# Multiresolution Tangent Distance for Affine-invariant Classification

**Nuno Vasconcelos**      **Andrew Lippman**
MIT Media Laboratory, 20 Ames St, E15-320M,
Cambridge, MA 02139, {nuno,lip}@media.mit.edu

## Abstract

The ability to rely on similarity metrics invariant to image transformations is an important issue for image classification tasks such as face or character recognition. We analyze an invariant metric that has performed well for the latter - the *tangent distance* - and study its limitations when applied to regular images, showing that the most significant among these (convergence to local minima) can be drastically reduced by computing the distance in a multiresolution setting. This leads to the *multiresolution tangent distance*, which exhibits significantly higher invariance to image transformations, and can be easily combined with robust estimation procedures.

## 1  Introduction

Image classification algorithms often rely on distance metrics which are too sensitive to variations in the imaging environment or set up (e.g. the Euclidean and Hamming distances), or on metrics which, even though less sensitive to these variations, are application specific or too expensive from a computational point of view (e.g. deformable templates).

A solution to this problem, combining invariance to image transformations with computational simplicity and general purpose applicability was introduced by Simard et al in [7]. The key idea is that, when subject to spatial transformations, images describe manifolds in a high dimensional space, and an invariant metric should measure the distance between those manifolds instead of the distance between other properties of (or features extracted from) the images themselves. Because these manifolds are complex, minimizing the distance between them is a difficult optimization problem which can, nevertheless, be made tractable by considering the minimization of the distance between the tangents to the manifolds -the *tangent distance* (TD) - instead of that between the manifolds themselves. While it has led to impressive results for the problem of character recognition [8], the linear approximation inherent to the TD is too stringent for regular images, leading to invariance over only a very narrow range of transformations.

In this work we embed the distance computation in a multiresolution framework [3], leading to the *multiresolution tangent distance* (MRTD). Multiresolution decompositions are common in the vision literature and have been known to improve the performance of image registration algorithms by extending the range over which linear approximations hold [5, 1]. In particular, the MRTD has several appealing properties: 1) maintains the general purpose nature of the TD; 2) can be easily combined with robust estimation procedures, exhibiting invariance to moderate non-linear image variations (such as caused by slight variations in shape or occlusions); 3) is amenable to computationally efficient screening techniques where bad matches are discarded at low resolutions; and 4) can be combined with several types of classifiers. Face recognition experiments show that the MRTD exhibits a significantly extended invariance to image transformations, originating improvements in recognition accuracy as high as 38%, for the hardest problems considered.

## 2   The tangent distance

Consider the manifold described by all the possible linear transformations that a pattern $I(\mathbf{x})$ may be subject to

$$T_{\mathbf{p}}\left[I(\mathbf{x})\right] = I(\psi(\mathbf{x}, \mathbf{p})), \tag{1}$$

where $\mathbf{x}$ are the spatial coordinates over which the pattern is defined, $\mathbf{p}$ is the set of parameters which define the transformation, and $\psi$ is a function typically linear on $\mathbf{p}$, but not necessarily linear on $\mathbf{x}$. Given two patterns $M(\mathbf{x})$ and $N(\mathbf{x})$, the distance between the associated manifolds - *manifold distance* (MD) - is

$$\mathcal{T}(M, N) = \min_{\mathbf{p}, \mathbf{q}} ||T_{\mathbf{q}}[M(\mathbf{x})] - T_{\mathbf{p}}[N(\mathbf{x})]||^2. \tag{2}$$

For simplicity, we consider a version of the distance in which only one of the patterns is subject to a transformation, i.e.

$$\mathcal{T}(M, N) = \min_{\mathbf{p}} ||M(\mathbf{x}) - T_{\mathbf{p}}[N(\mathbf{x})]||^2, \tag{3}$$

but all results can be extended to the two-sided distance. Using the fact that

$$\nabla_{\mathbf{p}} T_{\mathbf{p}}[N(\mathbf{x})] = \nabla_{\mathbf{p}} N(\psi(\mathbf{x}, \mathbf{p})) = \nabla_{\mathbf{p}} \psi(\mathbf{x}, \mathbf{p}) \nabla_{\mathbf{x}} N(\psi(\mathbf{x}, \mathbf{p})), \tag{4}$$

where $\nabla_{\mathbf{p}} T_{\mathbf{p}}$ is the gradient of $T_{\mathbf{p}}$ with respect to $\mathbf{p}$, $T_{\mathbf{p}}[N(\mathbf{x})]$ can, for small $\mathbf{p}$, be approximated by a first order Taylor expansion around the identity transformation

$$T_{\mathbf{p}}[N(\mathbf{x})] = N(\mathbf{x}) + (\mathbf{p} - \mathbf{I})^T \nabla_{\mathbf{p}} \psi(\mathbf{x}, \mathbf{p}) \nabla_{\mathbf{x}} N(\mathbf{x}).$$

This is equivalent to approximating the manifold by a tangent hyper-plane, and leads to the TD. Substituting this expression in equation 3, setting the gradient with respect to $\mathbf{p}$ to zero, and solving for $\mathbf{p}$ leads to

$$\mathbf{p} = \left[\sum_{\mathbf{x}} \nabla_{\mathbf{p}} \psi(\mathbf{x}, \mathbf{p}) \nabla_{\mathbf{x}} N(\mathbf{x}) \nabla_{\mathbf{x}}^T N(\mathbf{x}) \nabla_{\mathbf{p}}^T \psi(\mathbf{x}, \mathbf{p})\right]^{-1} \sum_{\mathbf{x}} D(\mathbf{x}) \nabla_{\mathbf{p}} \psi(\mathbf{x}, \mathbf{p}) \nabla_{\mathbf{x}} N(\mathbf{x}) + \mathbf{I}, \tag{5}$$

where $D(\mathbf{x}) = M(\mathbf{x}) - N(\mathbf{x})$. Given this optimal $\mathbf{p}$, the TD between the two patterns is computed using equations 1 and 3. The main limitation of this formulation is that it relies on a first-order Taylor series approximation, which is valid only over a small range of variation in the parameter vector $\mathbf{p}$.

### 2.1   Manifold distance via Newton's method

The minimization of the MD of equation 3 can also be performed through Newton's method, which consists of the iteration

$$\mathbf{p}^{n+1} = \mathbf{p}^n - \alpha \left[\nabla_{\mathbf{p}}^2 \mathcal{T}|_{\mathbf{p}=\mathbf{p}^n}\right]^{-1} \nabla_{\mathbf{p}} \mathcal{T}|_{\mathbf{p}=\mathbf{p}^n} \tag{6}$$

where $\nabla_{\mathbf{p}}\mathcal{T}$ and $\nabla_{\mathbf{p}}^2\mathcal{T}$ are, respectively, the gradient and Hessian of the cost function of equation 3 with respect to the parameter $\mathbf{p}$,

$$\nabla_{\mathbf{p}}\mathcal{T} = 2\sum_{\mathbf{x}}\left[M(\mathbf{x}) - T_{\mathbf{p}}[N(\mathbf{x})]\right]\nabla_{\mathbf{p}}T_{\mathbf{p}}[N(\mathbf{x})]$$

$$\nabla_{\mathbf{p}}^2\mathcal{T} = 2\sum_{\mathbf{x}}\left[-\nabla_{\mathbf{p}}T_{\mathbf{p}}[N(\mathbf{x})]\nabla_{\mathbf{p}}^T T_{\mathbf{p}}[N(\mathbf{x})] + \left[M(\mathbf{x}) - N(\mathbf{x})\right]\nabla_{\mathbf{p}}^2 T_{\mathbf{p}}[N(\mathbf{x})]\right].$$

Disregarding the term which contains second-order derivatives ($\nabla_{\mathbf{p}}^2 T_{\mathbf{p}}[N(\mathbf{x})]$), choosing $\mathbf{p}^0 = \mathbf{I}$ and $\alpha = 1$, using 4, and substituting in 6 leads to equation 5. I.e. the TD corresponds to a single iteration of the minimization of the MD by a simplified version of Newton's method, where second-order derivatives are disregarded. This reduces the rate of convergence of Newton's method, and a single iteration may not be enough to achieve the local minimum, even for simple functions. It is, therefore, possible to achieve improvement if the iteration described by equation 6 is repeated until convergence.

## 3   The multiresolution tangent distance

The iterative minimization of equation 6 suffers from two major drawbacks [2]: 1) it may require a significant number of iterations for convergence and 2), it can easily get trapped in local minima. Both these limitations can be, at least partially, avoided by embedding the computation of the MD in a multiresolution framework, leading to the *multiresolution manifold distance* (MRMD). For its computation, the patterns to classify are first subject to a multiresolution decomposition, and the MD is then iteratively computed for each layer, using the estimate obtained from the layer above as a starting point,

$$\mathbf{p}_l^{n+1} = \mathbf{p}_l^n + \alpha\left[\sum_{\mathbf{x}}\nabla_{\mathbf{p}}T_{\mathbf{p}_l^n}[N(\mathbf{x})]\nabla_{\mathbf{p}}^T T_{\mathbf{p}_l^n}[N(\mathbf{x})]\right]^{-1}\sum_{\mathbf{x}}D_l^n(\mathbf{x})\nabla_{\mathbf{p}}T_{\mathbf{p}_l^n}[N(\mathbf{x})], \quad (7)$$

where, $D_l^n(\mathbf{x}) = M(\mathbf{x}) - T_{\mathbf{p}_l^n}[N(\mathbf{x})]$. If only one iteration is allowed at each image resolution, the MRMD becomes the multiresolution extension of the TD, i.e. the *multiresolution tangent distance* (MRTD).

To illustrate the benefits of minimization over different scales consider the signal $f(t) = \sum_{k=1}^K \sin(w_k t)$, and the manifold generated by all its possible translations $f'(t, d) = f(t + d)$. Figure 1 depicts the multiresolution Gaussian decomposition of $f(t)$, together with the Euclidean distance to the points on the manifold as a function of the translation associated with each of them ($d$). Notice that as the resolution increases, the distance function has more local minima, and the range of translations over which an initial guess is guaranteed to lead to convergence to the global minimum (at $d = 0$) is smaller. I.e., at higher resolutions, a better initial estimate is necessary to obtain the same performance from the minimization algorithm.

Notice also that, since the function to minimize is very smooth at the lowest resolutions, the minimization will require few iterations at these resolutions if a procedure such as Newton's method is employed. Furthermore, since the minimum at one resolution is a good guess for the minimum at the next resolution, the computational effort required to reach that minimum will also be small. Finally, since a minimum at low resolutions is based on coarse, or global, information about the function or patterns to be classified, it is likely to be the global minimum of at least a significant region of the parameter space, if not the true global minimum.

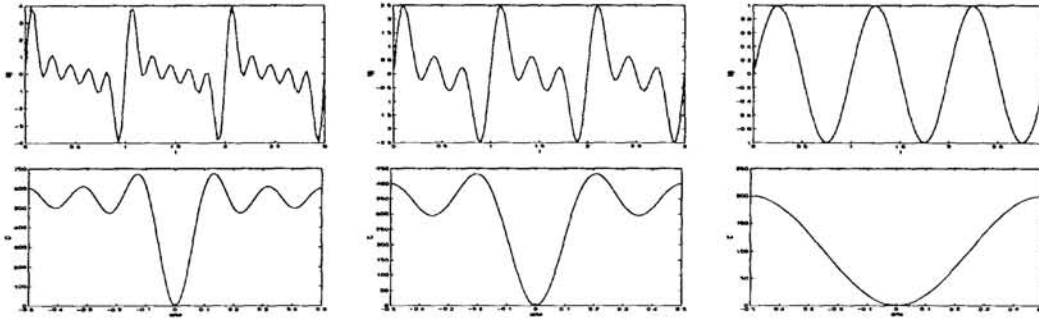

Figure 1: Top: Three scales of the multiresolution decomposition of $f(t)$. Bottom: Euclidean distance vs. translation for each scale. Resolution decreases from left to right.

## 4   Affine-invariant classification

There are many linear transformations which can be used in equation 1. In this work, we consider manifolds generated by affine transformations

$$\psi(\mathbf{x}, \mathbf{p}) = \begin{bmatrix} x & y & 1 & 0 & 0 & 0 \\ 0 & 0 & 0 & x & y & 1 \end{bmatrix} \mathbf{p} = \Phi(\mathbf{x})\mathbf{p}, \tag{8}$$

where $\mathbf{p}$ is the vector of parameters which characterize the transformation. Taking the gradient of equation 8 with respect to $\mathbf{p}$, $\nabla_{\mathbf{p}}\psi(\mathbf{x}, \mathbf{p}) = \Phi(\mathbf{x})^T$, using equation 4, and substituting in equation 7,

$$\begin{aligned} \mathbf{p}_l^{n+1} &= \mathbf{p}_l^n + \alpha \left[ \sum_{\mathbf{x}} \Phi(\mathbf{x})^T \nabla_{\mathbf{x}} N'(\mathbf{x}) \nabla_{\mathbf{x}}^T N'(\mathbf{x}) \Phi(\mathbf{x})^T \right]^{-1} \\ &\quad \sum_{\mathbf{x}} D'(\mathbf{x}) \Phi(\mathbf{x})^T \nabla_{\mathbf{x}} N'(\mathbf{x}), \end{aligned} \tag{9}$$

where $N'(\mathbf{x}) = N(\psi(\mathbf{x}, \mathbf{p}_l^n))$, and $D'(\mathbf{x}) = M(\mathbf{x}) - N'(\mathbf{x})$. For a given level $l$ of the multiresolution decomposition, the iterative process of equation 9 can be summarized as follows.

1. Compute $N'(\mathbf{x})$ by warping the pattern to classify $N(\mathbf{x})$ according to the best current estimate of $\mathbf{p}$, and compute its spatial gradient $\nabla_{\mathbf{x}} N'(\mathbf{x})$.

2. Update the estimate of $\mathbf{p}_l$ according to equation 9.

3. Stop if convergence, otherwise go to 1.

Once the final $\mathbf{p}_l$ is obtained, it is passed to the multiresolution level below (by doubling the translation parameters), where it is used as initial estimate. Given the values of $\mathbf{p}_i$ which minimize the MD between a pattern to classify and a set of prototypes in the database, a K-nearest neighbor classifier is used to find the pattern's class.

## 5   Robust classifiers

One issue of importance for pattern recognition systems is that of robustness to outliers, i.e errors which occur with low probability, but which can have large magnitude. Examples are errors due to variation of facial features (e.g. faces shot with or without glasses) in face recognition, errors due to undesired blobs of ink or uneven line thickness in character recognition, or errors due to partial occlusions (such as a hand in front of a face) or partially

missing patterns (such as an undoted *i*). It is well known that a few (maybe even one) outliers of high leverage are sufficient to throw mean squared error estimators completely off-track [6].

Several robust estimators have been proposed in the statistics literature to avoid this problem. In this work we consider *M-estimators* [4] which can be very easily incorporated in the MD classification framework. M-estimators are an extension of least squares estimators where the square function is substituted by a functional $\rho(x)$ which weighs large errors less heavily. The robust-estimator version of the tangent distance then becomes to minimize the cost function

$$\mathcal{T}(M, N) = \min_{\mathbf{P}} \sum_{\mathbf{x}} \rho(M(\mathbf{x}) - T_{\mathbf{P}}[N(\mathbf{x})]), \tag{10}$$

and it is straightforward to show that the "robust" equivalent to equation 9 is

$$\begin{aligned}
\mathbf{p}_l^{n+1} &= \mathbf{p}_l^n + \alpha \quad \left[ \sum_{\mathbf{x}} \rho''[D(\mathbf{x})] \Phi(\mathbf{x})^T \nabla_{\mathbf{x}} N'(\mathbf{x}) \nabla_{\mathbf{x}}^T N'(\mathbf{x}) \Phi(\mathbf{x})^T \right]^{-1} \times \\
&\quad \left[ \sum_{\mathbf{x}} \rho'[D(\mathbf{x})] \Phi(\mathbf{x})^T \nabla_{\mathbf{x}} N'(\mathbf{x}) \right],
\end{aligned} \tag{11}$$

where $D(\mathbf{x}) = M(\mathbf{x}) - N'(\mathbf{x})$ and $\rho'(x)$ and $\rho''(x)$ are, respectively, the first and second derivatives of the function $\rho(x)$ with respect to its argument.

## 6   Experimental results

In this section, we report on experiments carried out to evaluate the performance of the MD classifier. The first set of experiments was designed to test the invariance of the TD to affine transformations of the input. The second set was designed to evaluate the improvement obtained under the multiresolution framework.

### 6.1   Affine invariance of the tangent distance

Starting from a single view of a reference face, we created an artificial dataset composed by 441 affine transformations of it. These transformations consisted of combinations of all rotations in the range from $-30$ to 30 degrees with increments of 3 degrees, with all scaling transformations in the range from 70% to 130% with increments of 3%. The faces associated with the extremes of the scaling/rotation space are represented on the left portion of figure 2.

On the right of figure 2 are the distance surfaces obtained by measuring the distance associated with several metrics at each of the points in the scaling/rotation space. Five metrics were considered in this experiment: the Euclidean distance (ED), the TD, the MD computed through Newton's method, the MRMD, and the MRTD.

While the TD exhibits some invariance to rotation and scaling, this invariance is restricted to a small range of the parameter space and performance only slightly better than the obtained with the ED. The performance of the MD computed through Newton's method is dramatically superior, but still inferior to those achieved with the MRTD (which is very close to zero over the entire parameter space considered in this experiment), and the MRMD. The performance of the MRTD is in fact impressive given that it involves a computational increase of less than 50% with respect to the TD, while each iteration of Newton's method requires an increase of 100%, and several iterations are typically necessary to attain the minimum MD.

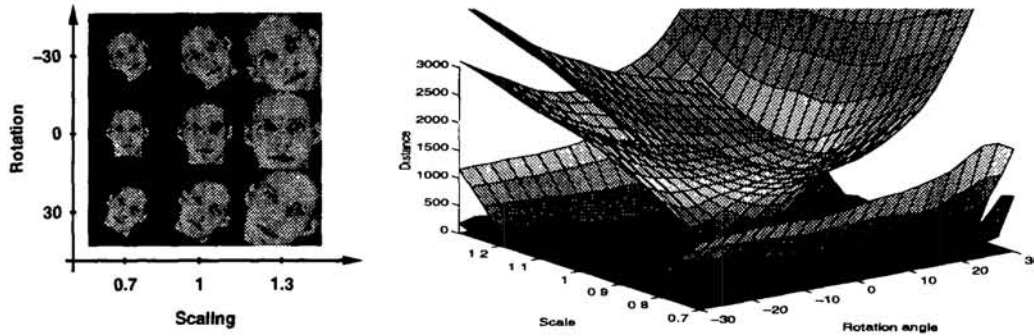

Figure 2: Invariance of the tangent distance. In the right, the surfaces shown correspond to ED, TD, MD through Newton's method, MRTD, and MRMD. This ordering corresponds to that of the nesting of the surfaces, i.e. the ED is the cup-shaped surface in the center, while the MRMD is the flat surface which is approximately zero everywhere.

## 6.2   Face recognition

To evaluate the performance of the multiresolution tangent distance on a real classification task, we conducted a series of face recognition experiments, using the Olivetti Research Laboratories (ORL) face database. This database is composed by 400 images of 40 subjects, 10 images per subject, and contains variations in pose, light conditions, expressions and facial features, but small variability in terms of scaling, rotation, or translation. To correct this limitation we created three artificial datasets by applying to each image three random affine transformations drawn from three multivariate normal distributions centered on the identity transformation with different covariances. A small sample of the faces in the database is presented in figure 3, together with its transformed version under the set of transformations of higher variability.

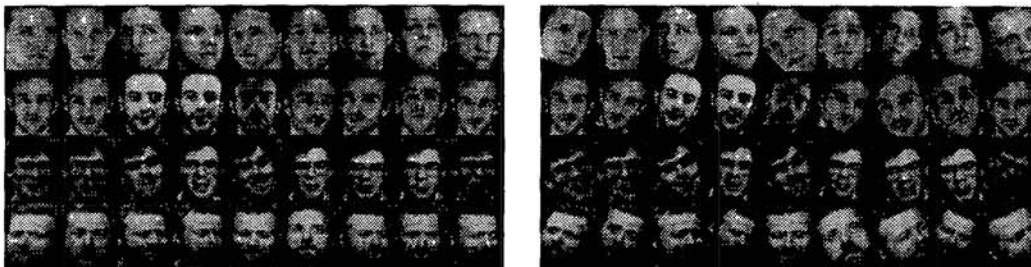

Figure 3: Left: sample of the ORL face database. Right: transformed version.

We next designed three experiments with increasing degree of difficulty. In the first, we selected the first view of each subject as the test set, using the remaining nine views as training data. In the second, the first five faces were used as test data while the remaining five were used for training. Finally, in the third experiment, we reverted the roles of the datasets used in the first. The recognition accuracy for each of these experiments and each of the datasets is reported on figure 4 for the ED, the TD, the MRTD, and a robust version of this distance (RMRTD) with $\rho(x) = \frac{1}{2}x^2$ if $x \leq \sigma T$ and $\rho(x) = \frac{T^2}{2}$ otherwise, where $T$ is a threshold (set to 2.0 in our experiments), and $\sigma$ a robust version of the error standard deviation defined as $\sigma = \text{median} \ |e_i - \text{median} \ (e_i)| \ / \ 0.6745$.

Several conclusions can be taken from this figure. First, it can be seen that the MRTD provides a significantly higher invariance to linear transformations than the ED or the TD,

increasing the recognition accuracy by as much as 37.8% in the hardest datasets. In fact, for the easier tasks of experiments one and two, the performance of the multiresolution classifier is almost constant and always above the level of 90% accuracy. It is only for the harder experiment that the invariance of the MRTD classifier starts to break down. But even in this case, the degradation is graceful - the recognition accuracy only drops below 75% for considerable values of rotation and scaling (dataset D3).

On the other hand, the ED and the single resolution TD break down even for the easier tasks, and fail dramatically when the hardest task is performed on the more difficult datasets. Furthermore, their performance does not degrade gracefully, they seem to be more invariant when the training set has five views than when it is composed by nine faces of each subject in the database.

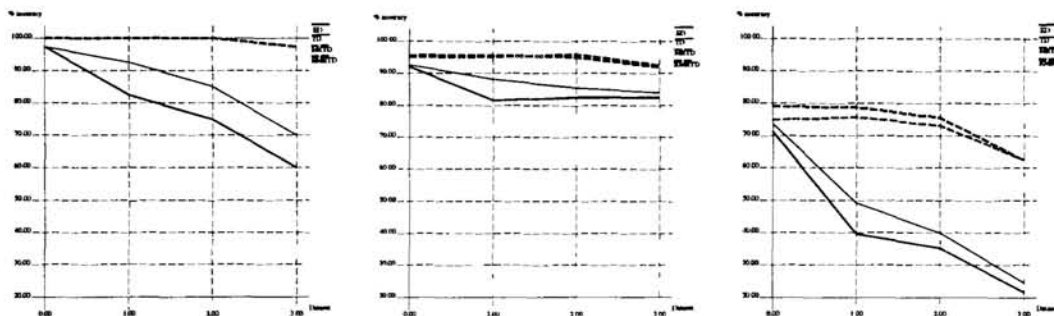

Figure 4: Recognition accuracy. From left to right: results from the first, second, and third experiments. Datasets are ordered by degree of variability: D0 is the ORL database D3 is subject to the affine transformations of greater amplitude.

## Acknowledgments

We would like to thank Federico Girosi for first bringing the tangent distance to our attention, and for several stimulating discussions on the topic.

## References

[1] P. Anandan, J. Bergen, K. Hanna, and R. Hingorani. Hierarchical Model-Based Motion Estimation. In M. Sezan and R. Lagendijk, editors, *Motion Analysis and Image Sequence Processing*, chapter 1. Kluwer Academic Press, 1993.

[2] D. Bertsekas. *Nonlinear Programming*. Athena Scientific, 1995.

[3] P. Burt and E. Adelson. The Laplacian Pyramid as a Compact Image Code. *IEEE Trans. on Communications*, Vol. 31:532–540, 1983.

[4] P. Huber. *Robust Statistics*. John Wiley, 1981.

[5] B. Lucas and T. Kanade. An Iterative Image Registration Technique with an Application to Stereo Vision. In *Proc. DARPA Image Understanding Workshop*, 1981.

[6] P. Rousseeuw and A. Leroy. *Robust Regression and Outlier Detection*. John Wiley, 1987.

[7] P. Simard, Y. Le Cun, and J. Denker. Efficient Pattern Recognition Using a New Transformation Distance. In *Proc. Neural Information Proc. Systems*, Denver, USA, 1994.

[8] P. Simard, Y. Le Cun, and J. Denker. Memory-based Character Recognition Using a Transformation Invariant Metric. In *Int. Conference on Pattern Recognition*, Jerusalem, Israel, 1994.
